# Multiplicative Updating Rule for Blind Separation Derived from the Method of Scoring

**Howard Hua Yang**
Department of Computer Science
Oregon Graduate Institute
PO Box 91000, Portland, OR 97291, USA
hyang@cse.ogi.edu

## Abstract

For blind source separation, when the Fisher information matrix is used as the Riemannian metric tensor for the parameter space, the steepest descent algorithm to maximize the likelihood function in this Riemannian parameter space becomes the serial updating rule with equivariant property. This algorithm can be further simplified by using the asymptotic form of the Fisher information matrix around the equilibrium.

## 1 Introduction

The relative gradient was introduced by (Cardoso and Laheld, 1996) to design multiplicative updating algorithms with equivariant property for blind separation problems. The idea is to calculate differentials by using a relative increment instead of an absolute increment in the parameter space. This idea has been extended to compute the relative Hessian by (Pham, 1996).

For a matrix function $f = f(W)$, the relative gradient is defined by

$$\widehat{\nabla} f = \frac{\partial f}{\partial W} W^T. \tag{1}$$

From the differential of $f(W)$ based on the relative gradient, the following learning rule is given by (Cardoso and Laheld, 1996) to maximize the function $f$:

$$\frac{dW}{dt} = \eta \widehat{\nabla} f W = \eta \frac{\partial f}{\partial W} W^T W \tag{2}$$

Also motivated by designing blind separation algorithms with equivariant property,

the natural gradient defined by

$$\tilde{\nabla} f = \frac{\partial f}{\partial W} W^T W \qquad (3)$$

was introduced in (Amari et al, 1996) which yields the same learning rule (2). The geometrical meaning of the natural gradient is given by (Amari, 1996). More details about the natural gradient can be found in (Yang and Amari, 1997) and (Amari, 1997).

The framework of the natural gradient learning was proposed by (Amari, 1997). In this framework, the ordinary gradient descent learning algorithm in the Euclidean space is not optimal in minimizing a function defined in a Riemannian space. The ordinary gradient should be replaced by the natural gradient which is defined by operating the inverse of the metric tensor in the Riemannian space on the ordinary gradient. Let $w$ denote a parameter vector. It is proved by (Amari, 1997) that if $C(w)$ is a loss function defined on a Riemannian space $\{w\}$ with a metric tensor $G$, the negative natural gradient of $C(w)$, namely, $-G^{-1}\frac{\partial C}{\partial w}$ is the steepest descent direction to decrease this function in the Riemannian space. Therefore, the steepest descent algorithm in this Riemannian space has the following form:

$$\frac{dw}{dt} = -\eta G^{-1} \frac{\partial C}{\partial w}.$$

If the Fisher information matrix is used as the metric tensor for the Riemannian space and $C(w)$ is replaced by the negative log-likelihood function, the above learning rule becomes the method of scoring ( Kay, 1993) which is the focus of this paper.

Both the relative gradient $\hat{\nabla}$ and the natural gradient $\tilde{\nabla}$ were proposed in order to design the multiplicative updating algorithms with the equivariant property. The former is due to a multiplicative increment in calculating differential while the latter is due to an increment based on a nonholonomic basis (Amari, 1997). Neither $\hat{\nabla}$ nor $\tilde{\nabla}$ depends on the data model. The Fisher information matrix is a special and important choice for the Riemannian metric tensor for statistical estimation problems. It depends on the data model. Operating the inverse of the Fisher information matrix on the ordinary gradient, we have another gradient operator. It is called a natural gradient induced by the Fisher information matrix.

In this paper, we show how to derive a multiplicative updating algorithm from the method of scoring. This approach is different from those based on the relative gradient and the natural gradient defined by (3).

## 2    Fisher Information Matrix For Blind Separation

Consider a linear mixing system:

$$x = As$$

where $A \in \Re^{n \times n}$, $x = (x_1, \cdots, x_n)^T$ and $s = (s_1, \cdots, s_n)^T$. Assume that sources are independent with a factorized joint pdf:

$$r(s) = \prod_{i=1}^{n} r(s_i).$$

The likelihood function is

$$p(x; A) = \frac{r(A^{-1}x)}{|A|}$$

where $|A| = |det(A)|$. Let $W = A^{-1}$ and $y = Wx$ ( a demixing system), then we have the log-likelihood function

$$L(W) = \sum_{i=1}^{n} \log r_i(y_i) + \log|W|.$$

It is easy to obtain

$$\frac{\partial L}{\partial w_{ij}} = \frac{r_i'(y_i)}{r_i(y_i)} x_j + W_{ij}^{-T} \tag{4}$$

where $W_{ij}^{-T}$ is the $(i, j)$ entry in $W^{-T} = (W^{-1})^T$. Writing (4) in a matrix form, we have

$$\frac{\partial L}{\partial W} = W^{-T} - \Phi(y)x^T = (I - \Phi(y)y^T)W^{-T} = F(y)W^{-T} \tag{5}$$

where $\Phi(y) = (\phi_1(y_1), \cdots, \phi_n(y_n))^T$, $\phi_i(y_i) = -\frac{r_i'(y_i)}{r_i(y_i)}$ and $F(y) = I - \Phi(y)y^T$.

The maximum likelihood algorithm based on the ordinary gradient $\frac{\partial L}{\partial W}$ is

$$\frac{dW}{dt} = \eta(I - \Phi(y)y^T)W^{-T} = \eta F(y)W^{-T}$$

which has the high computational complexity due to the matrix inverse $W^{-1}$. The maximum likelihood algorithm based on the natural gradient of matrix functions is

$$\frac{dW}{dt} = \eta \widetilde{\nabla} L = \eta(I - \Phi(y)y^T)W. \tag{6}$$

The same algorithm is obtained from $\frac{dW}{dt} = \eta \widehat{\nabla} L W$ by using the relative gradient. An apparent reason for using this algorithm is to avoid the matrix inverse $W^{-1}$. Another good reason for using it is due to the fact that the matrix $W$ driven by (6) never becomes singular if the initial matrix $W$ is not singular. This is proved by (Yang and Amari, 1997). In fact, this property holds for any learning rule of the following type:

$$\frac{dW}{dt} = H(y)W. \tag{7}$$

Let $<U, V> = \text{Tr}(U^T V)$ denote the inner product of $U$ and $V \in \Re^{n \times n}$. When $W(t)$ is driven by the equation (7), we have
$$\frac{d|W|}{dt} = <\frac{\partial|W|}{\partial W}, \frac{dW}{dt}> = <|W|(W^{-1})^T, \frac{dW}{dt}>$$
$$= \text{Tr}(|W|W^{-1}H(y)W) = \text{Tr}(H(y))|W|.$$

Therefore,

$$|W(t)| = |W(0)| \exp\{\int_0^t \text{Tr}(H(y(\tau)))d\tau\} \tag{8}$$

which is non-singular when the initial matrix $W(0)$ is non-singular.

The matrix function $F(y)$ is also called an estimating function. At the equilibrium of the system (6), it satisfies the zero condition $E[F(y)] = 0$, i.e.,

$$E[\phi_i(y_i)y_j] = \delta_{ij} \tag{9}$$

where $\delta_{ij} = 1$ if $i = j$ and 0 otherwise.

To calculate the Fisher information matrix, we need a vector form of the equation (5). Let Vec($\cdot$) denote an operator on a matrix which cascades the columns of the

matrix from the left to the right and forms a column vector. This operator has the following property:

$$\text{Vec}(\boldsymbol{ABC}) = (\boldsymbol{C}^T \otimes \boldsymbol{A})\text{Vec}(\boldsymbol{B}) \tag{10}$$

where $\otimes$ denotes the Kronecker product. Applying this property, we first rewrite (5) as

$$\frac{\partial L}{\partial \text{Vec}(\boldsymbol{W})} = \text{Vec}(\frac{\partial L}{\partial \boldsymbol{W}}) = (\boldsymbol{W}^{-1} \otimes \boldsymbol{I})\text{Vec}(\boldsymbol{F}(\boldsymbol{y})), \tag{11}$$

and then obtain the Fisher information matrix

$$\begin{aligned} \boldsymbol{G} &= E[\frac{\partial L}{\partial \text{Vec}(\boldsymbol{W})}(\frac{\partial L}{\partial \text{Vec}(\boldsymbol{W})})^T] \\ &= (\boldsymbol{W}^{-1} \otimes \boldsymbol{I})E[\text{Vec}(\boldsymbol{F}(\boldsymbol{y}))\text{Vec}^T(\boldsymbol{F}(\boldsymbol{y}))](\boldsymbol{W}^{-T} \otimes \boldsymbol{I}). \end{aligned} \tag{12}$$

The inverse of $\boldsymbol{G}$ is

$$\boldsymbol{G}^{-1} = (\boldsymbol{W}^T \otimes \boldsymbol{I})\boldsymbol{D}^{-1}(\boldsymbol{W} \otimes \boldsymbol{I}) \tag{13}$$

where $\boldsymbol{D} = E[\text{Vec}(\boldsymbol{F}(\boldsymbol{y}))\text{Vec}^T(\boldsymbol{F}(\boldsymbol{y}))]$.

## 3 Natural Gradient Induced By Fisher Information Matrix

Define a Riemannian space

$$\mathcal{V} = \{\text{Vec}(\boldsymbol{W}); \quad \boldsymbol{W} \in Gl(n)\}$$

in which the Fisher information matrix $\boldsymbol{G}$ is used as its metric. Here, $Gl(n)$ is the space of all the $n \times n$ invertible matrices.

Let $C(\boldsymbol{W})$ be a matrix function to be minimized. It is shown by (Amari, 1997) that the steepest descent direction in the Riemannian space $\mathcal{V}$ is $-\boldsymbol{G}^{-1}\frac{\partial C}{\partial \text{Vec}(\boldsymbol{W})}$.

Let us define the natural gradient in $\mathcal{V}$ by

$$\overline{\nabla}C(\boldsymbol{W}) = (\boldsymbol{W}^T \otimes \boldsymbol{I})\boldsymbol{D}^{-1}(\boldsymbol{W} \otimes \boldsymbol{I})\frac{\partial C}{\partial \text{Vec}(\boldsymbol{W})} \tag{14}$$

which is called the natural gradient induced by the Fisher information matrix. The time complexity of computing the natural gradient in the space $\mathcal{V}$ is high since inverting the matrix $\boldsymbol{D}$ of $n^2 \times n^2$ is needed.

Using the natural gradient in $\mathcal{V}$ to maximize the likelihood function $L(\boldsymbol{W})$ or the method of scoring, from (11) and (14) we have the following learning rule

$$\text{Vec}(\frac{d\boldsymbol{W}}{dt}) = \eta(\boldsymbol{W}^T \otimes \boldsymbol{I})\boldsymbol{D}^{-1}\text{Vec}(\boldsymbol{F}(\boldsymbol{y})) \tag{15}$$

We shall prove that the above learning rule has the equivariant property.

Denote $\text{Vec}^{-1}$ the inverse of the operator Vec. Let matrices $\boldsymbol{B}$ and $\boldsymbol{A}$ be of $n^2 \times n^2$ and $n \times n$, respectively. Denote $\boldsymbol{B}(i, \cdot)$ the i-th row of $\boldsymbol{B}$ and $\boldsymbol{B}_i = \text{Vec}^{-1}(\boldsymbol{B}(i, \cdot))$, $i = 1, \cdots, n^2$. Define an operator $\boldsymbol{B}\star$ as a mapping from $\Re^{n \times n}$ to $\Re^{n \times n}$:

$$\boldsymbol{B} \star \boldsymbol{A} = \begin{bmatrix} <\boldsymbol{B}_1, \boldsymbol{A}> & \cdots & <\boldsymbol{B}_{n^2-n+1}, \boldsymbol{A}> \\ \cdots & \cdots & \cdots \\ <\boldsymbol{B}_n, \boldsymbol{A}> & \cdots & <\boldsymbol{B}_{n^2}, \boldsymbol{A}> \end{bmatrix}$$

where $< \cdot, \cdot >$ is the inner product in $\Re^{n \times n}$. With the operation $\star$, we have

$$\boldsymbol{B}\text{Vec}(\boldsymbol{A}) = \begin{bmatrix} <\boldsymbol{B}_1, \boldsymbol{A}> \\ \vdots \\ <\boldsymbol{B}_{n^2}, \boldsymbol{A}> \end{bmatrix} = \text{Vec}(\text{Vec}^{-1}(\begin{bmatrix} <\boldsymbol{B}_1, \boldsymbol{A}> \\ \vdots \\ <\boldsymbol{B}_{n^2}, \boldsymbol{A}> \end{bmatrix})) = \text{Vec}(\boldsymbol{B} \star \boldsymbol{A}),$$

i.e.,

$$B\text{Vec}(\boldsymbol{A}) = \text{Vec}(\boldsymbol{B} \star \boldsymbol{A}).$$

Applying the above relation, we first rewrite the equation (15) as

$$\text{Vec}(\frac{d\boldsymbol{W}}{dt}) = \eta(\boldsymbol{W}^T \otimes \boldsymbol{I})\text{Vec}(\boldsymbol{D}^{-1} \star \boldsymbol{F}(\boldsymbol{y})),$$

then applying (10) to the above equation we obtain

$$\frac{d\boldsymbol{W}}{dt} = \eta(\boldsymbol{D}^{-1} \star \boldsymbol{F}(\boldsymbol{y}))\boldsymbol{W}. \tag{16}$$

**Theorem 1** *For the blind separation problem, the maximum likelihood algorithm based on the natural gradient induced by the Fisher information matrix or the method of scoring has the form (16) which is a multiplicative updating rule with the equivariant property.*

To implement the algorithm (16), we estimate $\boldsymbol{D}$ by sample average. Let $f_{ij}(\boldsymbol{y})$ be the $(i, j)$ entry in $\boldsymbol{F}(\boldsymbol{y})$. A general form for the entries in $\boldsymbol{D}$ is

$$d_{ij,kl} = E[f_{ij}(\boldsymbol{y})f_{kl}(\boldsymbol{y})]$$

which depends on the source pdfs $r_i(s_i)$. When the source pdfs are unknown, in practice we choose $r_i(s_i)$ as our prior assumptions about the source pdfs. To simplify the algorithm (16), we replace $\boldsymbol{D}$ by its asymptotic form at the solution points $\boldsymbol{a} = (c_1 s_{\sigma(1)}, \cdots, c_n s_{\sigma(n)})^T$ where $(\sigma(1), \cdots, \sigma(n))$ is a permutation of $(1, \cdots, n)$.

Regarding the structure of the asymptotic $\boldsymbol{D}$, we have the following theorem:

**Theorem 2** *Assume that the pdfs of the sources $s_i$ are even functions.*
*Then at the solution point $\boldsymbol{a} = (c_1 s_{\sigma(1)}, \cdots, c_n s_{\sigma(n)})^T$, $\boldsymbol{D}$ is a diagonal matrix and its $n^2$ diagonal entries have two forms, namely,*

$$E[f_{ij}(\boldsymbol{a})f_{ij}(\boldsymbol{a})] = \mu_i \lambda_j, \quad \text{for } i \neq j \text{ and}$$
$$E[(f_{ii}(\boldsymbol{a}))^2] = \nu_i$$

*where $\mu_i = E[\phi_i^2(a_i)]$, $\lambda_i = E[a_i^2]$ and $\nu_i = E[\phi_i^2(a_i)a_i^2] - 1$. More concisely, we have*

$$\boldsymbol{D} = diag(\text{Vec}(\boldsymbol{H})) \tag{17}$$

*where*

$$\boldsymbol{H} = (\mu_i \lambda_j)_{n \times n} - diag(\mu_1 \lambda_1, \cdots, \mu_n \lambda_n) + diag(\nu_1, \cdots, \nu_n)$$

The proof of Theorem 2 is given in Appendix 1.

Let $\boldsymbol{H} = (h_{ij})_{n \times n}$. Since all $\mu_i$, $\lambda_i$, and $\nu_i$ are positive, and so are all $h_{ij}$. We define

$$\frac{1}{\boldsymbol{H}} = (\frac{1}{h_{ij}})_{n \times n}.$$

Then from (17), we have

$$\boldsymbol{D}^{-1} = diag(\text{Vec}(\frac{1}{\boldsymbol{H}})).$$

The results in Theorem 2 enable us to simplify the algorithm (16) to obtain a low complexity learning rule. Since $\boldsymbol{D}^{-1}$ is a diagonal matrix, for any $n \times n$ matrix $\boldsymbol{A}$ we have

$$\boldsymbol{D}^{-1}\text{Vec}(\boldsymbol{A}) = \text{Vec}(\frac{1}{\boldsymbol{H}} \odot \boldsymbol{A}) \tag{18}$$

where $\odot$ denotes the componentwise multiplication of two matrices of the same dimension. Applying (18) to the learning rule (15), we obtain the following learning rule

$$\text{Vec}(\frac{dW}{dt}) = \eta(W^T \otimes I)\text{Vec}(\frac{1}{H} \odot F(y)).$$

Again, applying (10) to the above equation we have the following learning rule

$$\frac{dW}{dt} = \eta(\frac{1}{H} \odot F(y))W. \tag{19}$$

Like the learning rule (16), the algorithm (19) is also multiplicative; but unlike (16), there is no need to inverse the $n^2 \times n^2$ matrix in (19). The computation of $\frac{1}{H}$ is straightforward by computing the reciprocals of the entries in $H$.

$(\mu_i, \lambda_i, \nu_i)$ are $3n$ unknowns in $G$. Let us impose the following constraint

$$\nu_i = \mu_i \lambda_i. \tag{20}$$

Under this constraint, the number of unknowns in $G$ is $2n$, and $D$ can be written as

$$D = D_\lambda \otimes D_\mu \tag{21}$$

where $D_\lambda = \text{diag}(\lambda_1, \cdots, \lambda_n)$ and $D_\mu = \text{diag}(\mu_1, \cdots, \mu_n)$.

From (14), using (21) we have the natural gradient descent rule in the Riemannian space $\mathcal{V}$

$$\frac{d\text{Vec}(W)}{dt} = -\eta(W^T D_\lambda^{-1} W \otimes D_\mu^{-1})\frac{\partial C}{\partial \text{Vec}(W)}. \tag{22}$$

Applying the property (10), we rewrite the above equation in a matrix form

$$\frac{dW}{dt} = -\eta D_\mu^{-1} \frac{\partial C}{\partial W} W^T D_\lambda^{-1} W. \tag{23}$$

Since $\mu_i$ and $\lambda_i$ are unknown, $D_\mu$ and $D_\lambda$ are replaced by the identity matrix in practice. Therefore, the algorithm (2) is an approximation of the algorithm (23).

Taking $C = -L(W)$ as the negative likelihood function and applying the expression (5), we have the following maximum likelihood algorithm based on the natural gradient in $\mathcal{V}$:

$$\frac{dW}{dt} = \eta D_\mu^{-1}(I - \Phi(y)y^T)D_\lambda^{-1}W. \tag{24}$$

Again, replacing $D_\mu$ and $D_\lambda$ by the identity matrix we obtain the maximum likelihood algorithm (6) based on the relative gradient or natural gradient of matrix functions.

In the context of the blind separation, the source pdfs are unknown. The prior assumption $r_i(s_i)$ used to define the functions $\phi_i(y_i)$ may not match the true pdfs of the sources. However, the algorithm (24) is generally robust to the mismatch between the true pdfs and the pdfs employed by the algorithm if the mismatch is not too large. See (Cardoso, 1997) and ( Pham, 1996) for example.

## 4   Conclusion

In the context of blind separation, when the Fisher information matrix is used as the Riemannian metric tensor for the parameter space, maximizing the likelihood function in this Riemannian space based on the steepest descent method is the method of scoring. This method yields a multiplicative updating rule with the equivariant property. It is further simplified by using the asymptotic form of the Fisher information matrix around the equilibrium.

## 5    Appendix

**Appendix 1** *Proof of Theorem 2:*

By definition $f_{ij}(y) = \delta_{ij} - \phi_i(y_i)y_j$. At the equilibrium $a = (c_1 s_{\sigma(1)}, \cdots, c_n s_{\sigma(n)})^T$, we have $E[\phi_i(a_i)a_j] = 0$ for $i \neq j$ and $E[\phi_i(a_i)a_i] = 1$. So $E[f_{ij}(a)] = 0$. Since the source pdfs are even functions, we have $E[a_i] = 0$ and $E[\phi_i(a_i)] = 0$. Applying these equalities, it is not difficult to verify that

$$E[f_{ij}(a)f_{kl}(a)] = 0, \quad \text{for } (i,j) \neq (k,l). \tag{25}$$

So, $D$ is a diagonal matrix and

$$E[f_{ii}(a)f_{ii}(a)] = E[(1 - \phi_i(a_i)a_i)^2] = E[\phi_i^2(a_i)a_i^2] - 1,$$

$$E[f_{ij}(a)f_{ij}(a)] = E[\phi_i^2(a_i)a_j^2] = \mu_i \lambda_j$$

for $i \neq j$.
Q.E.D.

## References

[1] S. Amari. Natural gradient works efficiently in learning. *Accepted by Neural Computation*, 1997.

[2] S. Amari. Neural learning in structured parameter spaces – natural Riemannian gradient. In *Advances in Neural Information Processing Systems, 9, ed. M. C. Mozer, M. I. Jordan and T. Petsche, The MIT Press: Cambridge, MA.*, pages 127–133, 1997.

[3] S. Amari, A. Cichocki, and H. H. Yang. A new learning algorithm for blind signal separation. In *Advances in Neural Information Processing Systems, 8, eds. David S. Touretzky, Michael C. Mozer and Michael E. Hasselmo, MIT Press: Cambridge, MA.*, pages 757–763, 1996.

[4] J.-F. Cardoso. Infomax and maximum likelihood for blind source separation. *IEEE Signal Processing Letters*, April 1997.

[5] J.-F. Cardoso and B. Laheld. Equivariant adaptive source separation. *IEEE Trans. on Signal Processing*, 44(12):3017–3030, December 1996.

[6] S. M. Kay. *Fundamentals of Statistical Signal Processing: Estimation Theory*. PTR Prentice Hall, Englewood Cliffs, 1993.

[7] D. T. Pham. Blind separation of instantaneous mixture of sources via an ica. *IEEE Trans. on Signal Processing*, 44(11):2768–2779, November 1996.

[8] H. H. Yang and S. Amari. Adaptive on-line learning algorithms for blind separation: Maximum entropy and minimum mutual information. *Neural Computation*, 9(7):1457–1482, 1997.
